# Stochastic Neighbor Embedding

**Geoffrey Hinton and Sam Roweis**
Department of Computer Science, University of Toronto
10 King's College Road, Toronto, M5S 3G5 Canada
{*hinton,roweis*}*@cs.toronto.edu*

## Abstract

We describe a probabilistic approach to the task of placing objects, described by high-dimensional vectors or by pairwise dissimilarities, in a low-dimensional space in a way that preserves neighbor identities. A Gaussian is centered on each object in the high-dimensional space and the densities under this Gaussian (or the given dissimilarities) are used to define a probability distribution over all the potential neighbors of the object. The aim of the embedding is to approximate this distribution as well as possible when the same operation is performed on the low-dimensional "images" of the objects. A natural cost function is a sum of Kullback-Leibler divergences, one per object, which leads to a simple gradient for adjusting the positions of the low-dimensional images. Unlike other dimensionality reduction methods, this probabilistic framework makes it easy to represent each object by a mixture of widely separated low-dimensional images. This allows ambiguous objects, like the document count vector for the word "bank", to have versions close to the images of both "river" and "finance" without forcing the images of outdoor concepts to be located close to those of corporate concepts.

## 1   Introduction

Automatic dimensionality reduction is an important "toolkit" operation in machine learning, both as a preprocessing step for other algorithms (e.g. to reduce classifier input size) and as a goal in itself for visualization, interpolation, compression, etc. There are many ways to "embed" objects, described by high-dimensional vectors or by pairwise dissimilarities, into a lower-dimensional space. Multidimensional scaling methods[1] preserve dissimilarities between items, as measured either by Euclidean distance, some nonlinear squashing of distances, or shortest graph paths as with Isomap[2, 3]. Principal components analysis (PCA) finds a linear projection of the original data which captures as much variance as possible. Other methods attempt to preserve local geometry (e.g. LLE[4]) or associate high-dimensional points with a fixed grid of points in the low-dimensional space (e.g. self-organizing maps[5] or their probabilistic extension GTM[6]). All of these methods, however, require each high-dimensional object to be associated with only a single location in the low-dimensional space. This makes it difficult to unfold "many-to-one" mappings in which a single ambiguous object really belongs in several disparate locations in the low-dimensional space. In this paper we define a new notion of embedding based on probable neighbors. Our algorithm, Stochastic Neighbor Embedding (SNE) tries to place the objects in a low-dimensional space so as to optimally preserve neighborhood identity, and can be naturally extended to allow multiple *different* low-d images of each object.

## 2   The basic SNE algorithm

For each object, $i$, and each potential neighbor, $j$, we start by computing the asymmetric probability, $p_{ij}$, that $i$ would pick $j$ as its neighbor:

$$p_{ij} = \frac{\exp(-d_{ij}^2)}{\sum_{k \neq i} \exp(-d_{ik}^2)} \qquad (1)$$

The dissimilarities, $d_{ij}^2$, may be given as part of the problem definition (and need not be symmetric), or they may be computed using the scaled squared Euclidean distance ("affinity") between two high-dimensional points, $\mathbf{x}_i, \mathbf{x}_j$ :

$$d_{ij}^2 = \frac{||\mathbf{x}_i - \mathbf{x}_j||^2}{2\sigma_i^2} \qquad (2)$$

where $\sigma_i$ is either set by hand or (as in some of our experiments) found by a binary search for the value of $\sigma_i$ that makes the entropy of the distribution over neighbors equal to $\log k$. Here, $k$ is the effective number of local neighbors or "perplexity" and is chosen by hand.

In the low-dimensional space we also use Gaussian neighborhoods but with a fixed variance (which we set without loss of generality to be $\frac{1}{2}$) so the *induced* probability $q_{ij}$ that point $i$ picks point $j$ as its neighbor is a function of the low-dimensional *images* $\mathbf{y}_i$ of all the objects and is given by the expression:

$$q_{ij} = \frac{\exp(-||\mathbf{y}_i - \mathbf{y}_j||^2)}{\sum_{k \neq i} \exp(-||\mathbf{y}_i - \mathbf{y}_k||^2)} \qquad (3)$$

The aim of the embedding is to match these two distributions as well as possible. This is achieved by minimizing a cost function which is a sum of Kullback-Leibler divergences between the original ($p_{ij}$) and induced ($q_{ij}$) distributions over neighbors for each object:

$$C = \sum_i \sum_j p_{ij} \log \frac{p_{ij}}{q_{ij}} = \sum_i KL(P_i || Q_i) \qquad (4)$$

The dimensionality of the $\mathbf{y}$ space is chosen by hand (much less than the number of objects). Notice that making $q_{ij}$ large when $p_{ij}$ is small wastes some of the probability mass in the $q$ distribution so there is a cost for modeling a big distance in the high-dimensional space with a small distance in the low-dimensional space, though it is less than the cost of modeling a small distance with a big one. In this respect, SNE is an improvement over methods like LLE [4] or SOM [5] in which widely separated data-points can be "collapsed" as near neighbors in the low-dimensional space. The intuition is that while SNE emphasizes local distances, its cost function cleanly enforces both keeping the images of nearby objects nearby *and* keeping the images of widely separated objects relatively far apart.

Differentiating C is tedious because $\mathbf{y}_k$ affects $q_{ij}$ via the normalization term in Eq. 3, but the result is simple:

$$\frac{\partial C}{\partial \mathbf{y}_i} = 2 \sum_j (\mathbf{y}_i - \mathbf{y}_j)(p_{ij} - q_{ij} + p_{ji} - q_{ji}) \qquad (5)$$

which has the nice interpretation of a sum of forces pulling $\mathbf{y}_i$ toward $\mathbf{y}_j$ or pushing it away depending on whether $j$ is observed to be a neighbor more or less often than desired.

Given the gradient, there are many possible ways to minimize $C$ and we have only just begun the search for the best method. Steepest descent in which all of the points are adjusted in parallel is inefficient and can get stuck in poor local optima. Adding random jitter that decreases with time finds much better local optima and is the method we used for the examples in this paper, even though it is still quite slow. We initialize the embedding by putting all the low-dimensional images in random locations very close to the origin. Several other minimization methods, including annealing the perplexity, are discussed in sections 5&6.

# 3  Application of SNE to image and document collections

As a graphic illustration of the ability of SNE to model high-dimensional, near-neighbor relationships using only two dimensions, we ran the algorithm on a collection of bitmaps of handwritten digits and on a set of word-author counts taken from the scanned proceedings of NIPS conference papers. Both of these datasets are likely to have intrinsic structure in many fewer dimensions than their raw dimensionalities: 256 for the handwritten digits and 13679 for the author-word counts.

To begin, we used a set of 3000 digit bitmaps from the UPS database[7] with 600 examples from each of the five classes 0,1,2,3,4. The variance of the Gaussian around each point in the 256-dimensional raw pixel image space was set to achieve a perplexity of 15 in the distribution over high-dimensional neighbors. SNE was initialized by putting all the $\mathbf{y}_i$ in random locations very close to the origin and then was trained using gradient descent with annealed noise. Although SNE was given no information about class labels, it quite cleanly separates the digit groups as shown in figure 1. Furthermore, within each region of the low-dimensional space, SNE has arranged the data so that properties like orientation, skew and stroke-thickness tend to vary smoothly. For the embedding shown, the SNE cost function in Eq. 4 has a value of 6719 nats; with a uniform distribution across low-dimensional neighbors, the cost is $3000 \log_e(2999/15) = 15894$ nats. We also applied principal component analysis (PCA)[8] to the same data; the projection onto the first two principal components does not separate classes nearly as cleanly as SNE because PCA is much more interested in getting the large separations right which causes it to jumble up some of the boundaries between similar classes. In this experiment, we used digit classes that do not have very similar pairs like 3 and 5 or 7 and 9. When there are more classes and only two available dimensions, SNE does not as cleanly separate very similar pairs.

We have also applied SNE to word-document and word-author matrices calculated from the OCRed text of NIPS volume 0-12 papers[9]. Figure 2 shows a map locating NIPS authors into two dimensions. Each of the 676 authors who published more than one paper in NIPS vols. 0-12 is shown by a dot at the position $\mathbf{y}_i$ found by SNE; larger red dots and corresponding last names are authors who published six or more papers in that period. Distances $d_{ij}$ were computed as the norm of the difference between log aggregate author word counts, summed across all NIPS papers. Co-authored papers gave fractional counts evenly to all authors. All words occurring in six or more documents were included, except for stopwords giving a vocabulary size of 13649. (The `bow` toolkit[10] was used for part of the pre-processing of the data.) The $\sigma_i$ were set to achieve a local perplexity of $k = 25$ neighbors. SNE seems to have grouped authors by broad NIPS field: generative models, support vector machines, neuroscience, reinforcement learning and VLSI all have distinguishable localized regions.

# 4  A full mixture version of SNE

The clean probabilistic formulation of SNE makes it easy to modify the cost function so that instead of a single image, each high-dimensional object can have several different versions of its low-dimensional image. These alternative versions have mixing proportions that sum to 1. Image-version $b$ of object $i$ has location $\mathbf{y}_{i_b}$ and mixing proportion $\pi_{i_b}$. The low-dimensional neighborhood distribution for $i$ is a mixture of the distributions induced by each of its image-versions across all image-versions of a potential neighbor $j$:

$$q_{ij} = \sum_b \pi_{i_b} \sum_c \frac{\pi_{j_c} \exp(-\|\mathbf{y}_{i_b} - \mathbf{y}_{j_c}\|^2)}{\sum_k \sum_d \pi_{k_d} \exp(-\|\mathbf{y}_{i_b} - \mathbf{y}_{k_d}\|^2)} \tag{6}$$

In this multiple-image model, the derivatives with respect to the image locations $\mathbf{y}_{i_b}$ are straightforward; the derivatives w.r.t the mixing proportions $\pi_{i_b}$ are most easily expressed

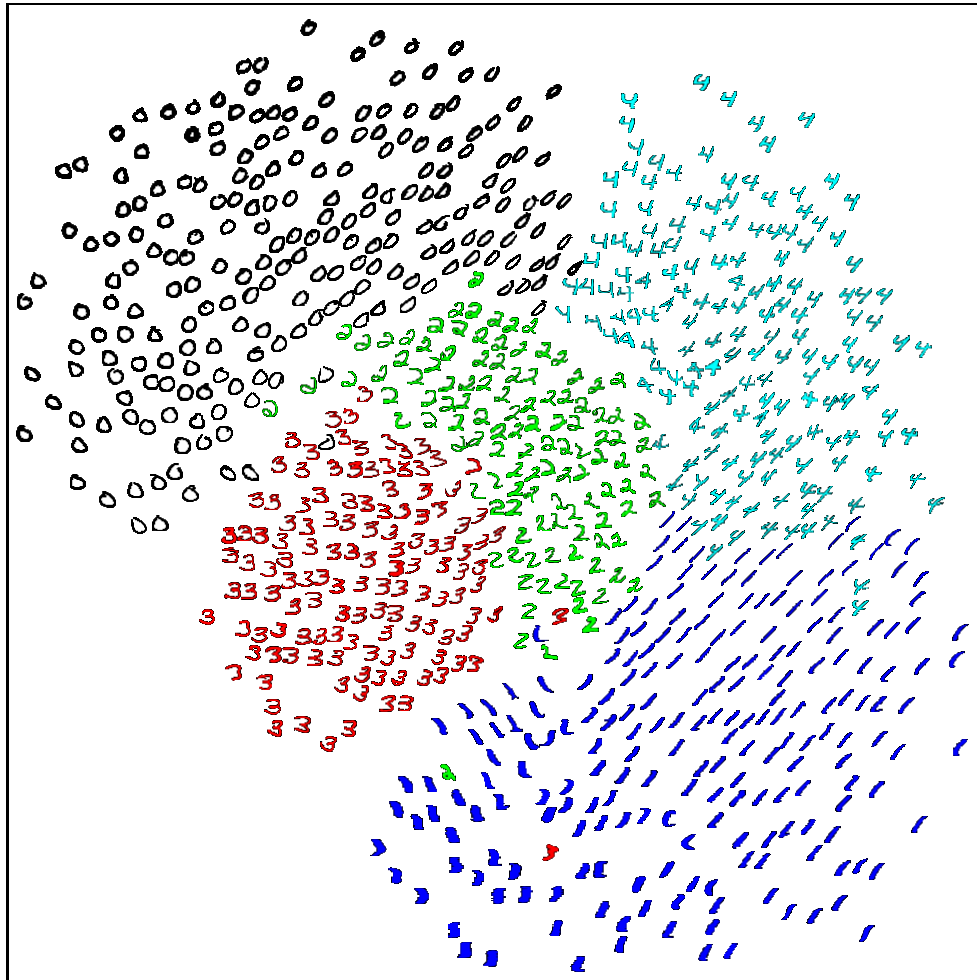

Figure 1: The result of running the SNE algorithm on 3000 256-dimensional grayscale images of handwritten digits. Pictures of the original data vectors $\mathbf{x}_i$ (scans of handwritten digit) are shown at the location corresponding to their low-dimensional images $\mathbf{y}_i$ as found by SNE. The classes are quite well separated even though SNE had no information about class labels. Furthermore, within each class, properties like orientation, skew and stroke-thickness tend to vary smoothly across the space. Not all points are shown: to produce this display, digits are chosen in random order and are only displayed if a 16 x 16 region of the display centered on the 2-D location of the digit in the embedding does not overlap any of the 16 x16 regions for digits that have already been displayed.

(SNE was initialized by putting all the $\mathbf{y}_i$ in random locations very close to the origin and then was trained using batch gradient descent (see Eq. 5) with annealed noise. The learning rate was 0.2. For the first 3500 iterations, each 2-D point was jittered by adding Gaussian noise with a standard deviation of 0.3 after each position update. The jitter was then reduced to 0 for a further 500 iterations.)

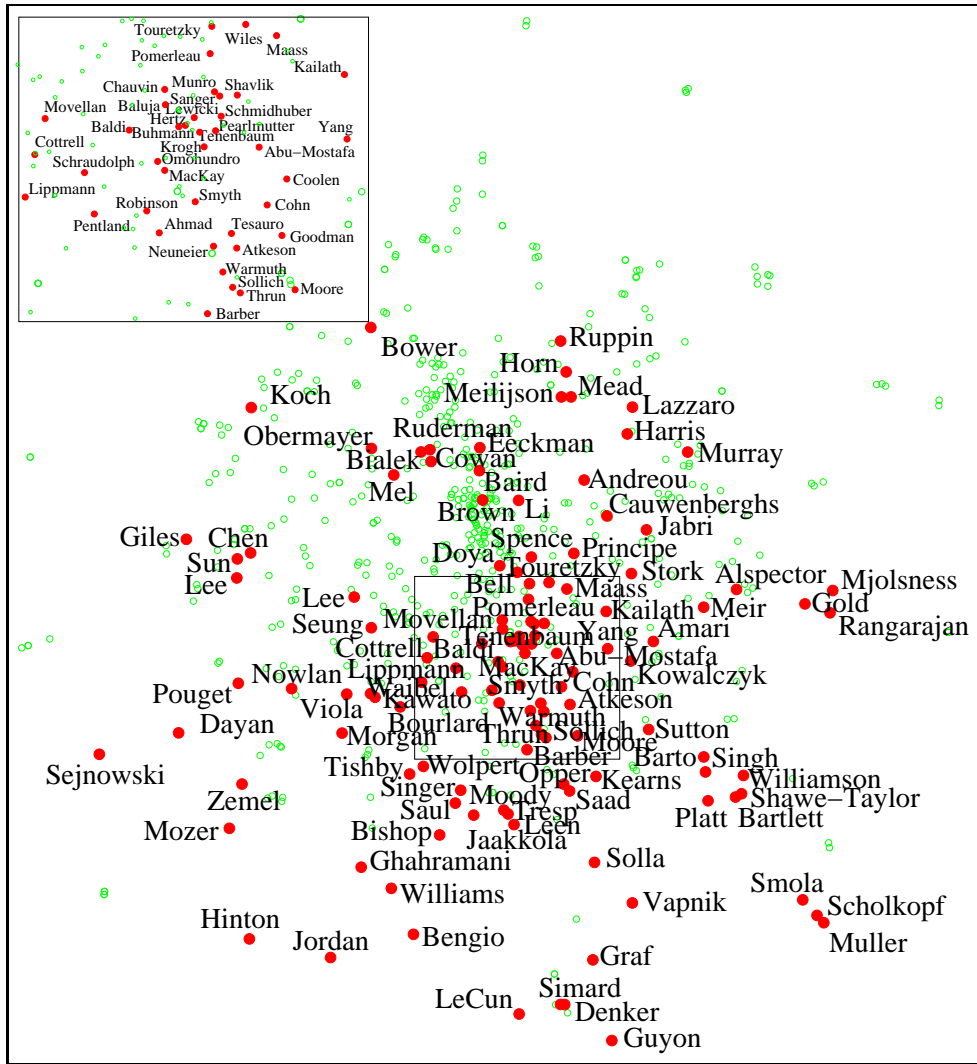

Figure 2: Embedding of NIPS authors into two dimensions. Each of the 676 authors who published more than one paper in NIPS vols. 0-12 is show by a dot at the location $\mathbf{y}_i$ found by the SNE algorithm. Larger red dots and corresponding last names are authors who published six or more papers in that period. The inset in upper left shows a blowup of the crowded boxed central portion of the space. Dissimilarities between authors were computed based on squared Euclidean distance between vectors of log aggregate author word counts. Co-authored papers gave fractional counts evenly to all authors. All words occurring in six or more documents were included, except for stopwords giving a vocabulary size of 13649. The NIPS text data is available at http://www.cs.toronto.edu/~roweis/data.html.

in terms of $r_{i_b j_c}$, the probability that version $b$ of $i$ picks version $c$ of $j$:

$$r_{i_b j_c} = \frac{\pi_{j_c} \exp(-||\mathbf{y}_{i_b} - \mathbf{y}_{j_c}||^2)}{\sum_k \sum_d \pi_{k_d} \exp(-||\mathbf{y}_{i_b} - \mathbf{y}_{k_d}||^2)} \tag{7}$$

The effect on $q_{ij}$ of changing the mixing proportion for version $g$ of object $m$ is given by

$$\frac{\partial q_{ij}}{\partial \pi_{m_g}} = \delta_{mi} \sum_c r_{m_g j_c} + \sum_b \frac{\pi_{i_b}}{\pi_{m_g}} r_{i_b m_g} \left[ \delta_{mj} - \sum_c r_{i_b j_c} \right] \tag{8}$$

where $\delta_{mi} = 1$ if $m = i$ and 0 otherwise. The effect of changing $\pi_{m_g}$ on the cost, C, is

$$\frac{\partial C}{\partial \pi_{m_g}} = -\sum_i \sum_j \frac{p_{ij}}{q_{ij}} \frac{\partial q_{ij}}{\partial \pi_{m_g}} \tag{9}$$

Rather than optimizing the mixing proportions directly, it is easier to perform unconstrained optimization on "softmax weights" defined by $\pi_{i_b} = \exp(w_{i_b})/\sum_f \exp(w_{i_f})$.

As a "proof-of-concept", we recently implemented a simplified mixture version in which every object is represented in the low-dimensional space by exactly two components that are constrained to have mixing proportions of $0.5$. The two components are pulled together by a force which increases linearly up to a threshold separation. Beyond this threshold the force remains constant.[1] We ran two experiments with this simplified mixture version of SNE. We took a dataset containing 300 pictures of each of the digits 2,3,4 and added 100 hybrid digit-pictures that were each constructed by picking new examples of two of the classes and taking each pixel at random from one of these two "parents". After mini-mization, 66% of the hybrids and only 19% of the non-hybrids had significantly different locations for their two mixture components. Moreover, the mixture components of each hybrid always lay in the regions of the space devoted to the classes of its two parents and never in the region devoted to the third class. For this example we used a perplexity of 10 in defining the local neighborhoods, a step size of for each position update of $0.7$ times the gradient, and used a constant jitter of $0.05$. Our very simple mixture version of SNE also makes it possible to map a circle onto a line without losing any near neighbor relationships or introducing any new ones. Points near one "cut point" on the circle can mapped to a mixture of two points, one near one end of the line and one near the other end. Obviously, the location of the cut on the two-dimensional circle gets decided by which pairs of mixture components split first during the stochastic optimization. For certain optimization param-eters that control the ease with which two mixture components can be pulled apart, only a single cut in the circle is made. For other parameter settings, however, the circle may fragment into two or more smaller line-segments, each of which is topologically correct but which may not be linked to each other.

The example with hybrid digits demonstrates that even the most primitive mixture version of SNE can deal with ambiguous high-dimensional objects that need to be mapped to two widely separated regions of the low-dimensional space. More work needs to be done before SNE is efficient enough to cope with large matrices of document-word counts, but it is the only dimensionality reduction method we know of that promises to treat homonyms sensibly without going back to the original documents to disambiguate each occurrence of the homonym.

# 5 Practical optimization strategies

Our current method of reducing the SNE cost is to use steepest descent with added jitter that is slowly reduced. This produces quite good embeddings, which demonstrates that the SNE cost function is worth minimizing, but it takes several hours to find a good embedding for just 3000 datapoints so we clearly need a better search algorithm.

The time per iteration could be reduced considerably by ignoring pairs of points for which all four of $p_{ij}, p_{ji}, q_{ij}, q_{ji}$ are small. Since the matrix $p_{ij}$ is fixed during the learning, it is natural to sparsify it by replacing all entries below a certain threshold with zero and renormalizing. Then pairs $i, j$ for which both $p_{ij}$ and $p_{ji}$ are zero can be ignored from gradient calculations if both $q_{ij}$ and $q_{ji}$ are small. This can in turn be determined in logarithmic time in the size of the training set by using sophisticated geometric data structures such as K-D trees, ball-trees and AD-trees, since the $q_{ij}$ depend only on $\|\mathbf{y}_i - \mathbf{y}_j\|^2$. Computational physics has attacked exactly this same complexity when performing multibody gravitational or electrostatic simulations using, for example, the fast multipole method.

In the mixture version of SNE there appears to be an interesting way of avoiding local optima that does not involve annealing the jitter. Consider two components in the mixture for an object that are far apart in the low-dimensional space. By raising the mixing proportion of one and lowering the mixing proportion of the other, we can move probability mass from one part of the space to another without it ever appearing at intermediate locations. This type of "probability wormhole" seems like a good way to avoid local optima that arise because a cluster of low-dimensional points must move through a bad region of the space in order to reach a better one.

Yet another search method, which we have used with some success on toy problems, is to provide extra dimensions in the low-dimensional space but to penalize non-zero values on these dimensions. During the search, SNE will use the extra dimensions to go around lower-dimensional barriers but as the penalty on using these dimensions is increased, they will cease to be used, effectively constraining the embedding to the original dimensionality.

# 6 Discussion and Conclusions

Preliminary experiments show that we can find good optima by first annealing the perplexities $\sigma_i^2$ (using high jitter) and only reducing the jitter after the final perplexity has been reached. This raises the question of what SNE is doing when the variance, $\sigma_i^2$, of the Gaussian centered on each high-dimensional point is very big so that the distribution across neighbors is almost uniform. It is clear that in the high variance limit, the contribution of $p_{ij} \log(p_{ij}/q_{ij})$ to the SNE cost function is just as important for distant neighbors as for close ones. When $\sigma_i^2$ is very large, it can be shown that SNE is equivalent to minimizing the mismatch between squared distances in the two spaces, provided all the squared distances from an object $i$ are first normalized by subtracting off their "antigeometric" mean, $g_i^2$:

$$\text{Mismatch} = \sum_{ij} \left[ (d_{ij}^2 - g_i^2) - (\hat{d}_{ij}^2 - \hat{g}_i^2) \right]^2 \tag{10}$$

$$d_{ij}^2 = \|\mathbf{x}_i - \mathbf{x}_j\|^2/\sigma^2, \qquad g_i^2 = -\log \sum_{k \neq i} \frac{\exp(-d_{ik}^2)}{n-1}, \tag{11}$$

$$\hat{d}_{ij}^2 = \|\mathbf{y}_i - \mathbf{y}_j\|^2/\sigma^2, \qquad \hat{g}_i^2 = -\log \sum_{k \neq i} \frac{\exp(-\hat{d}_{ik}^2)}{n-1} \tag{12}$$

where $n$ is the number of objects.

This mismatch is very similar to "stress" functions used in nonmetric versions of MDS, and enables us to understand the large-variance limit of SNE as a particular variant of such procedures. We are still investigating the relationship to metric MDS and to PCA.

SNE can also be seen as an interesting special case of Linear Relational Embedding (LRE) [11]. In LRE the data consists of triples (*e.g.* Colin has-mother Victoria) and the task is to predict the third term from the other two. LRE learns an N-dimensional vector for each object and an NxN-dimensional matrix for each relation. To predict the third term in a triple, LRE multiplies the vector representing the first term by the matrix representing the relationship and uses the resulting vector as the mean of a Gaussian. Its predictive distribution for the third term is then determined by the relative densities of all known objects under this Gaussian. SNE is just a degenerate version of LRE in which the only relationship is "near" and the matrix representing this relationship is the identity.

In summary, we have presented a new criterion, Stochastic Neighbor Embedding, for mapping high-dimensional points into a low-dimensional space based on stochastic selection of similar neighbors. Unlike self-organizing maps, in which the low-dimensional coordinates are fixed to a grid and the high-dimensional ends are free to move, in SNE the high-dimensional coordinates are fixed to the data and the low-dimensional points move. Our method can also be applied to arbitrary pairwise dissimilarities between objects if such are available instead of (or in addition to) high-dimensional observations. The gradient of the SNE cost function has an appealing "push-pull" property in which the forces acting on $\mathbf{y}_i$ to bring it closer to points it is under-selecting and further from points it is over-selecting as its neighbor. We have shown results of applying this algorithm to image and document collections for which it sensibly placed similar objects nearby in a low-dimensional space while keeping dissimilar objects well separated.

Most importantly, because of its probabilistic formulation, SNE has the ability to be extended to mixtures in which ambiguous high-dimensional objects (such as the word "bank") can have several widely-separated images in the low-dimensional space.

**Acknowledgments** We thank the anonymous referees and several visitors to our poster for helpful suggestions. Yann LeCun provided digit and NIPS text data. This research was funded by NSERC.

## Footnotes

[1]We used a threshold of $0.05$. At threshold the force was $0.025$ nats per unit length. The low-d space has a natural scale because the variance of the Gaussian used to determine $q_{ij}$ is fixed at $0.5$.

# References

[1] T. Cox and M. Cox. *Multidimensional Scaling*. Chapman & Hall, London, 1994.

[2] J. Tenenbaum. Mapping a manifold of perceptual observations. In *Advances in Neural Information Processing Systems*, volume 10, pages 682–688. MIT Press, 1998.

[3] J. B. Tenenbaum, V. de Silva, and J. C. Langford. A global geometric framework for nonlinear dimensionality reduction. *Science*, 290:2319–2323, 2000.

[4] S. T. Roweis and L. K. Saul. Nonlinear dimensionality reduction by locally linear embedding. *Science*, 290:2323–2326, 2000.

[5] T. Kohonen. *Self-organization and Associative Memory*. Springer-Verlag, Berlin, 1988.

[6] C. Bishop, M. Svensen, and C. Williams. GTM: The generative topographic mapping. *Neural Computation*, 10:215, 1998.

[7] J. J. Hull. A database for handwritten text recognition research. *IEEE Transaction on Pattern Analysis and Machine Intelligence*, 16(5):550–554, May 1994.

[8] I. T. Jolliffe. *Principal Component Analysis*. Springer-Verlag, New York, 1986.

[9] Yann LeCun. Nips online web site. http://nips.djvuzone.org, 2001.

[10] Andrew Kachites McCallum. Bow: A toolkit for statistical language modeling, text retrieval, classification and clustering. http://www.cs.cmu.edu/ mccallum/bow, 1996.

[11] A. Paccanaro and G.E. Hinton. Learning distributed representations of concepts from relational data using linear relational embedding. *IEEE Transactions on Knowledge and Data Engineering*, 13:232–245, 2000.
